# Connectionist Learning of Expert Preferences by Comparison Training

**Gerald Tesauro**

*IBM Thomas J. Watson Research Center*
*PO Box 704, Yorktown Heights, NY 10598 USA*

## Abstract

A new training paradigm, called the "comparison paradigm," is introduced for tasks in which a network must learn to choose a preferred pattern from a set of $n$ alternatives, based on examples of human expert preferences. In this paradigm, the input to the network consists of two of the $n$ alternatives, and the trained output is the expert's judgement of which pattern is better. This paradigm is applied to the learning of backgammon, a difficult board game in which the expert selects a move from a set of legal moves. With comparison training, much higher levels of performance can be achieved, with networks that are much smaller, and with coding schemes that are much simpler and easier to understand. Furthermore, it is possible to set up the network so that it always produces consistent rank-orderings.

## 1. Introduction

There is now widespread interest in the use of connectionist networks for real-world practical problem solving. The principal areas of application which have been studied so far involve relatively low-level signal processing and pattern recognition tasks. However, connectionist networks might also be useful in higher-level tasks which are currently tackled by expert systems and knowledge engineering approaches [2]. In this paper, we consider problem domains in which the expert is given a set of $n$ alternatives as input ($n$ may be either small or large), and must select the most desirable or most preferable alternative. This type of task occurs repeatedly throughout the domains of politics, business, economics, medicine, and many others. Whether it is choosing a foreign-policy option, a weapons contractor, a course of treatment for a disease, or simply what to have for dinner, problems requiring choice are constantly being faced and solved by human experts.

How might a learning system such as a connectionist network be set up to learn to make such choices from human expert examples? The immediately obvious approach is to train the network to produce a numerical output

"score" for each input alternative. To make a choice, then, one would have the network score each alternative, and select the alternative with the highest score. Since the learning system learns from examples, it seems logical to train the network on a data base of examples in which a human expert has entered a numerical score for each possible choice. However, there are two major problems with such an approach. First, in many domains in which $n$ is large, it would be tremendously time-consuming for the expert to create a data base in which each individual alternative has been painstaking evaluated, even the vast number of obviously bad alternatives which are not even worth considering. (It is important for the network to see examples of bad alternatives, otherwise it would tend to produce high scores for everything.) More importantly, in many domains human experts do not think in terms of absolute scoring functions, and it would thus be extremely difficult to create training data containing absolute scores, because such scoring is alien to the expert's way of thinking about the problem. Instead, the most natural way to make training data is simply to record the expert in action, i.e., for each problem situation, record each of the alternatives he had to choose from, and record which one he actually selected.

For these reasons, we advocate teaching the network to compare pairs of alternatives, rather than scoring individual alternatives. In other words, the input should be two of the set of $n$ alternatives, and the output should be a 1 or 0 depending on which of the two alternatives is better. From a set of recorded human expert preferences, one can then teach the network that the expert's choice is better than all other alternatives.

One potential concern raised by this approach is that, in performance mode after the network is trained, it might be necessary to make $n^2$ comparisons to select the best alternative, whereas only $n$ individual scores are needed in the other approach. However, the network can select the best alternative with only $n$ comparisons by going through the list of alternatives in order, and comparing the current alternative with the best alternative seen so far. If the current alternative is better, it becomes the new best alternative, and if it is worse, it is rejected. Another potential concern is that a network which only knows how to compare might not produce a consistent rank-ordering, i.e., it might say that alternative $a$ is better than $b$, $b$ is better than $c$, and $c$ is better than $a$, and then one does not know which alternative to select. However, we shall see later that it is possible to guarantee consistency with a constrained architecture which forces the network to come up with absolute numerical scores for individual alternatives.

In the following, we shall examine the application of the comparison training paradigm to the game of backgammon, as considerable experience has already been obtained in this domain. In previous papers [7,6], a network was described which learned to play backgammon from an expert data base, using the so-called "back-propagation" learning rule [5]. In that system, the network was trained to score individual moves. In other words, the input

consists of a move (defined by the initial position before the move and the final position after the move), and the desired output is a real number indicating the strength of the move. Henceforth we shall refer to this training paradigm as the "relative score" paradigm. While this approach produced considerable success, it had a number of serious limitations. We shall see that the comparison paradigm solves one of the most important limitations of the previous approach, with the result that the overall performance of the network is much better, the number of connections required is greatly reduced, and the network's input coding scheme is much simpler and easier to understand.

## 2.   Previous backgammon networks

In [7], a network was described which learned to play fairly good backgammon by back-propagation learning of a large expert training set, using the relative score paradigm described previously. After training, the network was tested both by measuring its performance on a test set of positions not used in training, and by actual game play against humans and conventional computer programs. The best network was able to defeat Sun Microsystems' *Gammontool*, the best available commercial program, by a substantial margin, but it was still far from human expert-level performance.

The basic conclusion of [7] was that it was possible to achieve decent levels of performance by this network learning procedure, but it was not an easy matter, and required substantial human intervention. The choice of a coding scheme for the input information, for example, was found to be an extremely important issue. The best coding schemes contained a great deal of domain-specific information. The best encoding of the "raw" board information was in terms of concepts that human experts use to describe local transitions, such as "slotting," "stripping," etc.. Also, a few "pre-computed features" were required in addition to the raw board information. Thus it was necessary to be a domain expert in order to design a suitable network coding scheme, and it seemed that the only way to discover the best coding scheme was by painstaking trial and error. This was somewhat disappointing, as it was hoped that the network learning procedure would automatically produce an expert backgammon network with little or no human effort.

## 3.   Comparison paradigm network set-up

In the standard practice of back-propagation, a comparison paradigm network would have an input layer, one or more layers of hidden units, and an output layer, with full connectivity between adjacent layers. The input layer would represent two final board positions $a$ and $b$, and the output layer would have just a single unit to represent which board position was better. The

teacher signal for the output unit would be a 1 if board position $a$ was better than $b$, and a 0 if $b$ was better than $a$.

The proposed comparison paradigm network would overcome the limitation of only being able to consider individual moves in isolation, without knowledge of what other alternatives are available. In addition, the sophisticated coding scheme that was developed to encode transition information would not be needed, since comparisons could be based solely on the final board states. The comparison approach offers greater sensitivity in distinguishing between close alternatives, and as stated previously, it corresponds more closely to the actual form of human expert knowledge.

These advantages are formidable, but there are some important problems with the approach as currently described. One technical problem is that the learning is significantly slower. This is because $2n$ comparisons per training position are presented to the network, where $n \sim 20$, whereas in the relative score approach, only about 3-4 moves per position would be presented. It was therefore necessary to develop a number of technical tricks to increase the speed of the simulator code for this specific application (to be described in a future publication).

A more fundamental problem with the approach, however, is the issue of *consistency* of network comparisons. Two properties are required for complete consistency: (1) The comparison between any two positions must be *unambiguous*, i.e., if the network says that $a$ is better than $b$ when $a$ is presented on the left and $b$ on the right, it had better say that $a$ is better than $b$ if $a$ is on the right and $b$ is on the left. One can show that this requires the network's output to exactly invert whenever the input board positions are swapped. (2) The comparisons must be *transitive*, as alluded to previously, i.e., if $a$ is judged better than $b$, and $b$ is judged better than $c$, the network had better judge $a$ to be better than $c$.

Standard unconstrained networks have no guarantee of satisfying either of these properties. After some thought, however, one realizes that the output inversion symmetry can be enforced by a symmetry relation amongst the weights in the network, and that the transitivity and rank-order consistency can be guaranteed by *separability* in the architecture, as illustrated in Figure 1. Here we see that this network really consists of two half-networks, one of which is only concerned with the evaluation of board position $a$, and the other of which is concerned only with the evaluation of board position $b$. (Due to the indicated symmetry relation, one needs only store one half-network in the simulator code.) Each half-network may have one or more layers of hidden units, but as long as they are not cross-coupled, the evaluation of each of the two input board positions is boiled down to a single real number. Since real numbers always rank-order consistently, the network's comparisons are always consistent.

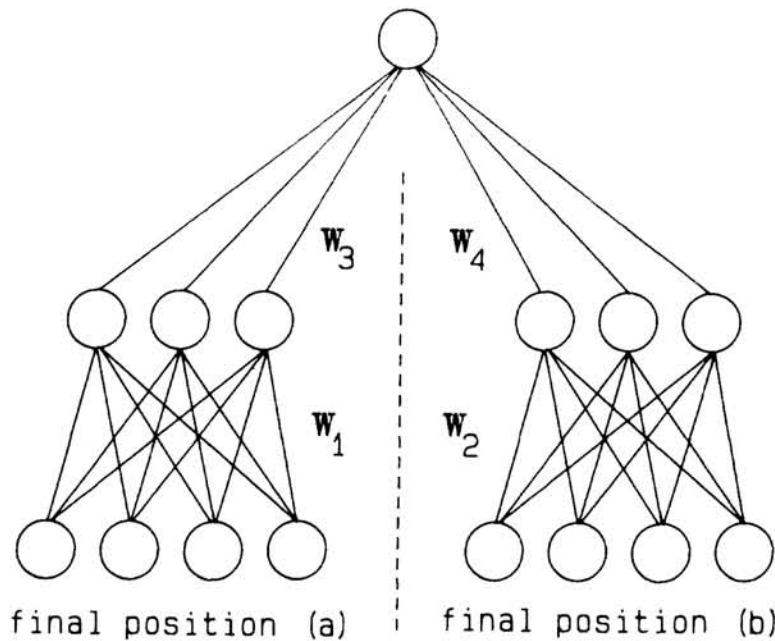

Figure 1: A network design for comparison training with guaranteed consistency of comparisons. Weight groups have symmetry relations $\mathbf{W_1} = \mathbf{W_2}$ and $\mathbf{W_3} = -\mathbf{W_4}$, which ensures that the output exactly inverts upon swapping positions in the input array. Separation of the hidden units condenses the evaluation of each final board position into a single real number, thus ensuring transitivity.

An important added benefit of this scheme is that an absolute board evaluation function is obtained in each half-network. This means that the network, to the extent that its evaluation function is accurate, has an intrinsic understanding of a given position, as opposed to merely being able to detect features which correspond to good moves. As has been emphasized by Berliner [1], an intrinsic understanding of the position is crucial for play at the highest levels, and for use of the doubling cube. Thus, this approach can serve as the basis for future progress, whereas the previous approach of scoring moves was doomed eventually to run into a dead end.

## 4.    Results of comparison training

The training procedure for the comparison paradigm network was as follows: Networks were set up with 289 input units which encode a description of single final board position, varying numbers of hidden units, and a single output unit. The training data was taken from a set of 400 games in which the author played both sides. This data set contains a recording of the author's preferred move for each position, and no other comments. The engaged positions in the data set were selected out (disengaged racing positions were not studied) and divided into five categories: bearoff, bearin, opponent bearoff, opponent bearin, and a default category covering everything else. In each

| Type of test set | RSP net (651-12-1) | CP net (289-1) |
|---|---|---|
| bearoff | .82 | .83 |
| bearin | .54 | .60 |
| opp. bearoff | .56 | .54 |
| opp. bearin | .60 | .66 |
| other | .58 | .65 |

Table 1: Performance of nets of indicated size on respective test sets, as measured by fraction of positions for which net agrees with human expert choice of best move. RSP: relative score paradigm, CP: comparison paradigm.

category, 200 positions chosen at random were set aside to be used as testing data; the remaining data (about 1000 positions in each category except the default category, for which about 4000 positions were used) was used to train networks which specialized in each category. The learning algorithm used was standard back-propagation with momentum and without weight decay.

Performance after training is summarized in Tables 1 and 2. Table 1 gives the performance of each specialist network on the appropriate set of test positions. Results for the comparison paradigm network are shown for networks without hidden units, because it was found that the addition of hidden units did not improve the performance. (This is discussed in the following section.) We contrast these results with results of training networks in the relative score paradigm on the same training data sets. We see in Table 1 that for the bearoff and opponent bearoff specialists, there is only a small change in performance under the comparison paradigm. For the bearin and opponent bearin specialists, there is an improvement in performance of about 6 percentage points in each case. For this particular application, this is a very substantial improvement in performance. However, the most important finding is for the default category, which is much larger and more difficult than any of the specialist categories. The default network's performance is the key factor in determining the system's overall game performance. With comparison training, we find an improvement in performance from 58% to 65%. Given the size and difficulty of this category, this can only be described as a huge improvement in performance, and is all the more remarkable when one considers that the comparison paradigm net has only 300 weights, as opposed to 8000 weights for the relative score paradigm net.

Next, a combined game-playing system was set up using the five specialist nets for all engaged positions. (The Gammontool evaluation function was called for racing positions.) Results are given in Table 2. Against Gammontool itself, the performance under the comparison paradigm improves from 59% to 64%. Against the author (and teacher), the performance improves from an estimated 35% (since the RSP nets are so big and slow, accurate

| Opponent | RSP nets | CP nets |
|---|---|---|
| Gammontool | .59 (500 games) | .64 (2000 games) |
| Tesauro | .35 (100 games) | .42 (400 games) |

Table 2: Game-playing performance of composite network systems against Gammontool and against the author, as measured by fraction of games won, without counting gammons or backgammons.

statistics could not be obtained) to about 42%.

Qualitatively, one notices a substantial overall improvement in the new network's level of play. But what is most striking is the network's *worst case* behavior. The previous relative-score network had particularly bad worst-case behavior: about once every other game, the network would make an atrocious blunder which would seriously jeopardize its chances of winning that game [6]. An alarming fraction of these blunders were seemingly random and could not be logically explained. The new comparison paradigm network's worst-case behavior is vastly improved in this regard. The frequency and severity of its mistakes are significantly reduced, but more importantly, its mistakes are understandable. (Some of the improvement in this respect may be due to the elimination of the noisy teacher signal described in [7].)

## 5.  Conclusions

We have seen that, in the domain of backgammon, the introduction of the comparison training paradigm has resulted in networks which perform much better, with vastly reduced numbers of weights, and with input coding schemes that are much simpler and easier to understand. It was surprising that such high performance could be obtained in "perceptron" networks, i.e., networks without hidden units. This reminds us that one should not summarily dismiss perceptrons as uninteresting or unworthy of study because they are only capable of learning linearly separable functions [3]. A substantial component of many difficult real-world problems may lie in the linearly separable spectrum, and thus it makes sense to try perceptrons at least as a first attempt. It was also surprising that the use of hidden units in the comparison-trained networks does not improve the performance. This is unexplained, and is the subject of current research. It is, however, not without precedent: in at least one other real-world application [4], it has been found that networks with hidden units do not perform any better than networks without hidden units.

More generally, one might conclude that, in training a neural network (or indeed any learning system) from human expert examples in a complex domain, there should be a good match between the natural form of the expert's knowledge and the method by which the network is trained. For domains in which the expert must select a preferred alternative from a set of alternatives,

the expert naturally thinks in terms of comparisons amongst the top few alternatives, and the comparison paradigm proposed here takes advantage of that fact. It would be possible in principle to train a network using absolute evaluations, but the creation of such a training set might be too difficult to undertake on a large scale.

If the above discussion is correct, then the comparison paradigm should be useful in other applications involving expert choice, and in other learning systems besides connectionist networks. Typically expert systems are hand-crafted by knowledge engineers, rather than learned from human expert examples; however, there has recently been some interest in supervised learning approaches. It will be interesting to see if the comparison paradigm proves to be useful when supervised learning procedures are applied to other domains involving expert choice. In using the comparison paradigm, it will be important to have some way to guarantee that the system's comparisons will be unambiguous and transitive. For feed-forward networks, it was shown in this paper how to guarantee this using symmetric, separated networks; it should be possible to impose similar constraints on other learning systems to enforce consistency.

## References

[1] H. Berliner, "On the construction of evaluation functions for large domains," *Proc. of IJCAI* (1979) 53–55.

[2] S. I. Gallant, "Connectionist expert systems," *Comm. ACM* **31**, 152–169 (1988).

[3] M. Minsky and S. Papert, *Perceptrons*, MIT Press, Cambridge MA (1969).

[4] N. Qian and T. J. Sejnowski, "Predicting the secondary structure of globular proteins using neural network models," *J. Mol. Biol.* **202**, 865–884 (1988).

[5] D. E. Rumelart and J. L. McClelland (eds.), *Parallel Distributed Processing: Explorations in the Microstructure of Cognition*, Vols. 1 and 2, MIT Press, Cambridge MA (1986).

[6] G. Tesauro, "Neural network defeats creator in backgammon match." Univ. of Illinois, Center for Complex Systems Technical Report CCSR-88-6 (1988).

[7] G. Tesauro and T. J. Sejnowski, "A parallel network that learns to play backgammon," *Artificial Intelligence*, in press (1989).
